# Green's Function Method for Fast On-line Learning Algorithm of Recurrent Neural Networks

**Guo-Zheng Sun, Hsing-Hen Chen and Yee-Chun Lee**
Institute for Advanced Computer Studies
and
Laboratory for Plasma Research,
University of Maryland
College Park, MD 20742

## Abstract

The two well known learning algorithms of recurrent neural networks are the back-propagation (Rumelhart & et. al., Werbos) and the forward propagation (Williams and Zipser). The main drawback of back-propagation is its off-line backward path in time for error cumulation. This violates the on-line requirement in many practical applications. Although the forward propagation algorithm can be used in an on-line manner, the annoying drawback is the heavy computation load required to update the high dimensional sensitivity matrix ($O(N^4)$ operations for each time step). Therefore, to develop a fast forward algorithm is a challenging task. In this paper we proposed a forward learning algorithm which is one order faster (only $O(N^3)$ operations for each time step) than the sensitivity matrix algorithm. The basic idea is that instead of integrating the high dimensional sensitivity dynamic equation we solve forward in time for its Green's function to avoid the redundant computations, and then update the weights whenever the error is to be corrected.

A Numerical example for classifying state trajectories using a recurrent network is presented. It substantiated the faster speed of the proposed algorithm than the Williams and Zipser's algorithm.

## I. Introduction.

In order to deal with sequential signals, recurrent neural networks are often put forward as a useful model. A particularly pressing issue concerning recurrent networks is the search for an efficient on-line training algorithm. Error back-propagation method (Rumelhart, Hinton, and Williams[1]) was originally proposed to handle feedforward networks. This method can be applied to train recurrent networks if one unfolds the time sequence of mappings into a multilayer feed-forward net, each layer with identical weights. Due to the nature of backward path, it is basically an off-line method. Pineda [2] generalized it to recurrent networks with hidden neurons. However, he is mostly interested in time-independent fixed point type of behaviors. Pearlmutter [3] proposed a scheme to learn temporal trajectories which involves equations to be solved backward in time. It is essentially a generalized version of error back-propagation to the problem of learning a target state trajectory. The viable on-line method to date is the RTRL (Real Time Recurrent Learning) algorithm (Williams and Zipser [4]), which propagates a sen-

sitivity matrix forward in time. The main drawback of this algorithm is its high cost of computation. It needs $O(N^4)$ number of operations per time step. Therefore, a faster (less than $O(N^4)$ operations) on-line algorithm appears to be desirable.

Toomarian and Barhen [5] proposed an $O(N^2)$ on-line algorithm. They derived the same equations as Pearlmutter's back-propagation using adjoint-operator approach. They then tried to convert the backward path into a forward path by adding a *Delta* function to its source term. But this is not correct. The problem is not merely because it "precludes straightforward numerical implementation" as they acknowledged later [6]. Even in theory, the result is not correct. The mistake is in their using a not well defined equity of the *Delta* function integration. Briefly speaking, the equity $\int_{t_0}^{t_f} \delta(t - t_f) f(t)\, dt = f(t_f)$ is not right if the function $f(t)$ is discontinuous at $t = t_f$. The value of the left-side integral depends on the distribution of function $f(t)$ and therefore is not uniquely defined. If we deal with the discontinuity carefully by splitting time interval from $t_0$ to $t_f$ into two segments: $t_0$ to $t_f\text{-}\varepsilon$ and $t_f\text{-}\varepsilon$ to $t_f$ and let $\varepsilon \to 0$, we will find out that adding a *Delta* function to the source term does not affect the basic property of the adjoint equation. Namely, it still has to be solved backward in time.

Recently, Toomarian and Barhen [6] modified their adjoint-operator approach and proposed an alternative $O(N^3)$ on-line training algorithm. Although, in nature, their result is very similar to what we presented in this paper, it will be seen that our approach is more straightforward and can be easily implemented numerically.

Schmidhuber[7] proposed an $O(N^3)$ algorithm which is a combination of back propagation (within each data block of size $N$) and forward propagation (between blocks). It is therefore not truly an on-line algorithm.

Sun, Chen and Lee [8] studied this problem, using a more general approach - variational approach, in which a constrained optimization problem with Lagrangian multipliers was considered. The dynamic equation of the Lagrangian multiplier was derived, which is exactly the same as adjoint equation[5]. By taking advantage of linearity of this equation an $O(N^3)$ on-line algorithm was derived. But, the numerical implementation of the algorithm, especially the numerical instabilities are not addressed in the paper.

In this paper we will present a new approach to this problem - the Green's function method. The advantages of the this method are the simple mathematical formulation and easy numerical implementation. One numerical example of trajectory classification is presented to substantiate the faster speed of the proposed algorithm. The numerical results are benchmarked with Williams and Zipser's algorithm.

## II. Green's Function Approach.

### (a) Definition of the Problem

Consider a fully recurrent network with neural activity represented by an N-dimensional vector x(t). The dynamic equations can be written in general as a set of first order differential equations:

$$\dot{x}(t) = F(x(t), w, I(t)) \qquad (1)$$

where $w$ is a matrix representing the set of weights and all other adjustable parameters, $I(t)$ is a vector representing the neuron units clamped by external input signals at time $t$. For a simple network connected by first order weights the nonlinear function $F$ may look like

$$F(x(t), w, I(t)) = -x(t) + g(w \cdot x) + I(t) \qquad (2)$$

where the scaler function $g(u)$ could be, for instance, the *Sigmoid* function $g(u) = 1/(1+e^{-u})$. Suppose that part of the state neurons $\{x_i \mid i \in M\}$ are measurable and part of neurons $\{x_i \mid i \in$

$H$} are hidden. For the measurable units we may have desired output $\hat{x}(t)$. In order to train the network, an objective functional (or an error measure functional) is often given to be

$$E(x, \hat{x}) = \int_{t_o}^{t_f} e(x(t), \hat{x}(t)) \, dt \tag{3}$$

where functional $E$ depends on weights $w$ implicitly through the measurable neurons $\{x_i \mid i \in M\}$. A typical error function *is*

$$e(x(t), \hat{x}(t)) = (x(t) - \hat{x}(t))^2 \tag{4}$$

The gradient descent learning is to modify the weights according to

$$\Delta w \propto -\frac{\partial E}{\partial w} = -\int_{t_o}^{t_f} \frac{\partial e}{\partial x} \cdot \frac{\partial x}{\partial w} \, dt. \tag{5}$$

In order o evaluate the integral in Eq. (5) one needs to know both $\partial e/\partial w$ and $\partial x/\partial w$. The first term can be easily obtained by taking derivative of the given error function $e(x(t), \hat{x}(t))$. For the second term one needs to solve the differential equation

$$\frac{d}{dt}(\frac{\partial x}{\partial w}) = \frac{\partial F}{\partial x} \cdot \frac{\partial x}{\partial w} + \frac{\partial F}{\partial w} \tag{6}$$

which is easily derived by taking derivative of Eq.(1) with respect to $w$. The well known forward algorithm of recurrent networks [4] is to solve Equation (6) forward in time and make the weight correction at the end ($t = t_f$) of the input sequence. (This algorithm was developed independently by several researchers, but due to the page limitation we could not refer all related papers and now simply call it Williams and Zipser's algorithm) The on-line learning is to make weight correction whenever an error is to be corrected during the input sequence

$$\Delta w(t) = -\eta \, (\frac{\partial e}{\partial x} \cdot \frac{\partial x}{\partial w}) \tag{7}$$

The proof of convergence of on-line learning algorithm will be addressed elsewhere.

The main drawback of this forward algorithm is that it requires $O(N^4)$ operations per time step to update the matrix $\partial x/\partial w$. Our goal of the Green's function approach is to find an on-line algorithm which requires less computation load.

## (b). Green's Function Solution

First let us analyze the computational complexity when integrating Eq. (6) directly. Rewrite Eq. (6) as

$$L \cdot \frac{\partial x}{\partial w} = \frac{\partial F}{\partial w} \tag{8}$$

where the linear operator $L$ is defined as $L = \frac{d}{dt} - \frac{\partial F}{\partial x}$

Two types of redundancy will be seen from Eq. (8). First, the operator $L$ does not depend on $w$ explicitly, which means that what we did in solving for $\partial x/\partial w$ is to repeatedly solve the identical differential equation for each components of $w$. This is redundant. It is especially wasteful when higher order connection weights are used. The second redundancy is in the special form of $\partial F/\partial w$ for neural computations where the same activity function (say, *Sigmoid* function) is

used for every neuron, so that

$$\frac{\partial F_k}{\partial w_{ij}} = g'\left(\sum_l w_{kl} \cdot x_l\right) \delta_{ki} \, x_j \tag{9}$$

where $\delta_{ki}$ is the Kronecker delta function. It is seen from Eq. (9) that among $N^3$ components of this third order tensor most of them, $N^2(N-1)$, are zero (when $k \neq i$) and need not to be computed repeatedly. In the original forward learning scheme, we did not pay attention to this redundancy.

Our Green's function approach is able to avoid the redundancy by solving for the low dimensional Green's function. And then we construct the solution of Eq. (8) by the dot product of $\partial F / \partial w$ with the Green's function, which can in turn be reduced to a scaler product due to Eq. (9).

The Green's function of the operator $L$ is defined as a dual time tensor function $G(t-\tau)$ which satisfies the following equation

$$\frac{d}{dt} G(t-\tau) - \frac{\partial F}{\partial x} \cdot G(t-\tau) = \delta(t-\tau) \tag{10}$$

It is well known that, if the solution of Eq. (10) is known, the solution of the original equation Eq. (6) (or (8)) can be constructed using the source term $\partial F / \partial w$ through the integral

$$\frac{\partial x}{\partial w}(t) = \int_{t_o}^{t_f} \left( G(t-\tau) \cdot \frac{\partial F}{\partial w}(\tau) \right) d\tau \tag{11}$$

To find the Green's function solution we first introduce a tensor function $V(t)$ that satisfies the homogeneous form of Eq. (10)

$$\begin{cases} \dfrac{d}{dt} V(t) - \dfrac{\partial F}{\partial x} \cdot V(t) = 0 \\ V(t_0) = 1 \end{cases} \tag{12}$$

The solution of Eq. (10) or the Green's function can then be constructed as

$$G(t-\tau) = V(t) \cdot V^1(\tau) H(t-\tau) \tag{13}$$

where $H(t\text{-}\tau)$ is the *Heaviside* function defined as

$$H(t-\tau) = \begin{cases} 1 & t \geq \tau \\ 0 & t < \tau \end{cases}$$

Using the well known equalities

$$\frac{d}{dt} H(t-\tau) = \delta(t-\tau)$$

and

$$f(t,\tau)\delta(t-\tau) = f(t,t)\delta(t-\tau) \quad,$$

one can easily verify that the constructed Green's function shown in Eq. (13) is correct, that is, it satisfies Eq. (10). Substituting $G(t\text{-}\tau)$ from Eq. (13) into Eq. (11) we obtain the solution of Eq. (6) as,

$$\frac{\partial x}{\partial w}(t) = V(t) \cdot \int_{t_o}^{t} \left( (V(\tau))^{-1} \cdot \frac{\partial F}{\partial w}(\tau) \right) d\tau \tag{14}$$

We note that this formal solution not only satisfies Eq. (6) but also satisfies the required initial condition

$$\frac{\partial x}{\partial w}(t_0) = 0 .$$

(15)

The "on-line" weight correction at time $t$ is obtained easily from Eq. (5)

$$\delta w = -\eta \frac{\partial e}{\partial x} \cdot \frac{\partial x}{\partial w} = -\eta \left( \frac{\partial e}{\partial x} \cdot V(t) \int_{t_o}^{t} ((V(\tau))^{-1} \cdot \frac{\partial F}{\partial w}(\tau)) d\tau \right)$$

(16)

## (c) Implementation

To implement Eq. (16) numerically we will introduce two auxiliary memories. First, we define $U(t)$ to be the inverse of matrix $V(t)$, i.e. $U(t) = V^{-1}(t)$. It is easy to see that the dynamic equation of $U(t)$ is

$$\begin{cases} \frac{d}{dt}U(t) + U(t) \cdot \frac{\partial F}{\partial x} = 0 \\ U(t_0) = 1 \end{cases}$$

(17)

Secondly, we define a third order tensor $\Pi_{ijk}$ that satisfies

$$\begin{cases} \frac{d\Pi}{dt} = U(t) \cdot \frac{\partial F}{\partial w} \\ \Pi(t_0) = 0 \end{cases}$$

(18)

then the weight correction in Eq. (16) becomes

$$\delta w = -\eta (v(t) \cdot \Pi(t))$$

(19)

where the vector $v(t)$ is the solution of the linear equation

$$v(t) \cdot U(t) = \frac{\partial e}{\partial x}$$

(20)

In discrete time, Eqs. (17) - (20) become:

$$\begin{cases} U_{ij}(t) = U_{ij}(t-1) + \Delta t \sum_{k} U_{ik}(t-1) \frac{\partial F_k}{\partial x_j} \\ U_{ij}(0) = \delta_{ij} \end{cases}$$

(21)

$$\begin{cases} \Pi_{ijk}(t) = \Pi_{ijk}(t-1) + (\Delta t) U_{ij}(t-1) \frac{\partial F_j}{\partial w_{jk}} \\ \Pi_{ijk}(0) = 0 \end{cases}$$

(22)

$$\sum_{i} v_i(t) \cdot U_{ij}(t) = \frac{\partial e}{\partial x_j}$$

(23)

$$\Delta w_{ij} = -\eta \left( \sum_k v_k(t) \Pi_{kij}(t) \right) \qquad (24)$$

To summarize the procedure of the Green's function method, we need to simultaneously integrate Eq. (21) and Eq. (22) for $U(t)$ and $\Pi$ forward in time starting from $U_{ij}(0) = \delta_{ij}$ and $\Pi_{ijk}(0) = 0$. Whenever error message are generated, we shall solve Eq. (23) for $v(t)$ and update weights according to Eq. (24).

The memory size required by this algorithm is simply $N^3 + N^2$ for storing $U(t)$ and $\Pi(t)$.

The speed of the algorithm is analyzed as follows. From Eq. (21) and Eq. (22) we see that the update of $U(t)$ and $\Pi$ both need $N^3$ operations per time step. To solve for $v(t)$ and update $w$, we need also $N^3$ operations per time step. So, the on-line updating of weights needs totally $4N^3$ operations per time step. This is one order of magnitude faster than the current forward learning scheme.

## III Numerical Simulation

We present in this section numerical examples to demonstrate the proposed learning algorithm and benchmark it against Williams&Zipser's algorithm.

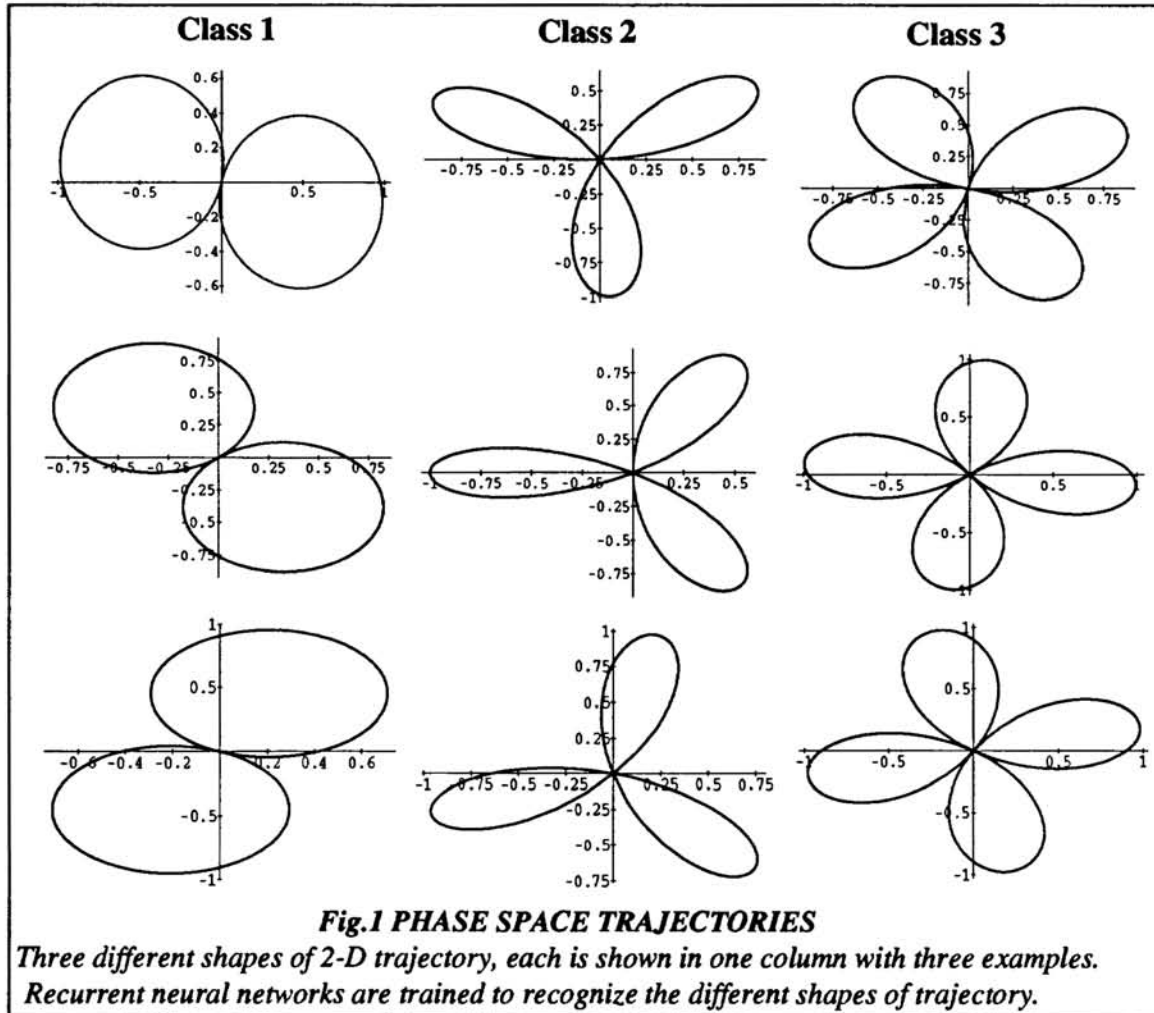

*Fig.1 PHASE SPACE TRAJECTORIES*
*Three different shapes of 2-D trajectory, each is shown in one column with three examples.*
*Recurrent neural networks are trained to recognize the different shapes of trajectory.*

We consider the trajectory classification problem. The input data are the time series of two

dimensional coordinate pairs $\{x(t), y(t)\}$ sampled along three different types of trajectories in the phase space. The sampling is taken uniformly with $\Delta t = 2\pi/60$. The trajectory equations are

$$
\begin{cases}
x(t) = \sin(t+\beta)\,|\sin(t)| \\
y(t) = \cos(t+\beta)\,|\sin(t)|
\end{cases}
\begin{cases}
x(t) = \sin(0.5t+\beta)\sin(1.5t) \\
y(t) = \cos(0.5t+\beta)\sin(1.5t)
\end{cases}
\begin{cases}
x(t) = \sin(t+\beta)\sin(2t) \\
y(t) = \cos(t+\beta)\sin(2t)
\end{cases}
$$

where $\beta$ is a uniformly distributed random parameter. When $\beta$ is changed, these trajectories are distorted accordingly. Nine examples (three for each class) are shown in Fig.1. The neural net used here is a fully recurrent first-order network with dynamics

$$
S_i(t+1) = S_i(t) + \left( Tanh\left( \sum_{j=1}^{N+6} W_{ij}(S \oplus I)_j \right) \right) \tag{25}
$$

where $S$ and $I$ are vectors of state and input neurons, the symbol $\oplus$ represents concatenation, and $N$ is the number of state. Six input neurons are used to represent the normalized vector $\{1, x(t), y(t), x(t)^2, y(t)^2, x(t)y(t)\}$. The neural network structure is shown in Fig. 2.

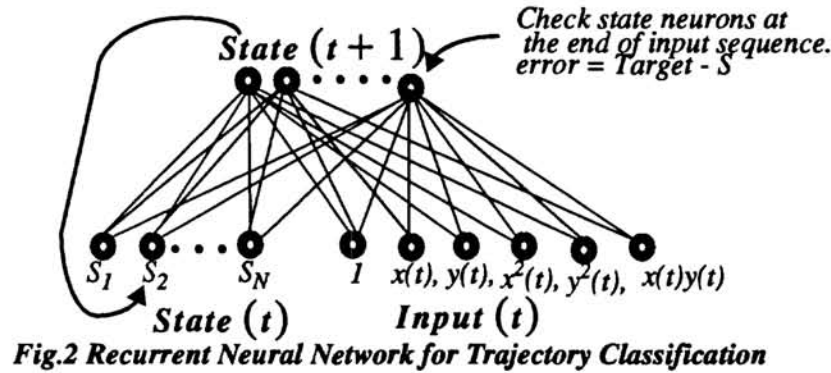

**Fig.2 Recurrent Neural Network for Trajectory Classification**

For recognition, each trajectory data sequence needs to be fed to the input neurons and the state neurons evolve according to the dynamics in Eq. (25). At the end of input series we check the last three state neurons and classify the input trajectory according to the "winner-take-all" rule. For training, we assign the desired final output for the three trajectory classes to (1,0,0), (0,1,0) and (0,0,1) respectively. Meanwhile, we need to simultaneously integrate Eq. (21) for $U(t)$ and Eq. (22) for $\Pi$. At the end, we calculated the error from Eq. (4) and solve Eq. (23) for $v(t)$ using LU decomposition algorithm. Finally, we update weights according to Eq. (24). Since the classification error is generated at the end of input sequence, this learning does not have to be on-line. We present this example only to compare the speeds of the proposed fast algorithm against the Williams and Zipser's. We run the two algorithms for the same number of iterations and compare the CPU time used. The results are shown in Table.1, where in each one iteration we present 150 training patterns, 50 for each class. These patterns are chosen by randomly selecting $\beta$ values. It is seen that the CPU time ratio is $O(1/N)$, indicating the Green's. function algorithm is one order faster in $N$.

Another issue to be considered is the error convergent rate (or learning rate, as usually called). Although the two algorithms calculate the same weight correction as in Eq. (7), due to different numerical schemes the outcomes may be different. As the result, the error convergent rates are slightly different even if the same learning rate $\eta$ is used. In all numerical simulations we have conducted the learning results are very good (in testing, the recognition is perfect, no single misclassification was found). But, during training the error convergence rates are different. The numerical experiments show that the proposed fast algorithm converges slower than

the Williams and Zipser's for the small size neural nets but faster for the large size neural net.

| Algorithm Simulation | Fast Algorithm | Williams&Zipser's | ratio |
|---|---|---|---|
| N = 4 (Number of Iterations = 200) | 1607.4 | 5020.8 | 1 : 3 |
| N = 8 (Number of Iterations = 50) | 1981.7 | 10807.0 | 1 : 5 |
| N = 12 (Number of Iterations = 50) | 5947.6 | 45503.0 | 1 : 8 |

*Table 1. The CPU time (in seconds) comparison, implemented in DEC3100 Workstation, for learning the trajectory classification example.*

## IV. Conclusion

The Green's function has been used to develop a faster on-line learning algorithm for recurrent neural networks. This algorithm requires $O(N^3)$ operations for each time step, which is one order faster than the Williams and Zipser's algorithm. The memory required is $O(N^3)$.

One feature of this algorithm is its straightforward formula, which can be easily implemented numerically. A numerical example of trajectory classification has been used to demonstrate the speed of this fast algorithm compared to Williams and Zipser's algorithm.

## References

[1] D.Rumelhart, G. Hinton, and R. Williams. Learning internal representations by error propagation. In Parallel distributed processing: Vol.I MIT press 1986. P. Werbos, Beyond Regression: New tools for prediction and analysis in the behavior sciences. Ph.D. thesis, Harvard university, 1974.

[2] F. Pineda, Generalization of back-propagation to recurrent neural networks. *Phys. Rev. Letters, 19(59):2229*, 1987.

[3] B. Pearlmutter, Learning state space trajectories in recurrent neural networks. *Neural Computation,1(2):263*, 1989.

[4] R. Williams and D. Zipser, A learning algorithm for continually running fully recurrent neural networks. Tech. Report ICS Report 8805, UCSD, La Jolla, CA 92093, November 1988.

[5] N. Toomarian, J. Barhen and S. Gulati, "Application of Adjoint Operators to Neural Learning", *Appl. Math. Lett., 3(3), 13-18*, 1990.

[6] N. Toomarian and J. Barhen, "Adjoint-Functions and Temporal Learning Algorithms in Neural Networks", *Advances in Neural Information Processing Systems 3*, p. 113-120, Ed. by R. Lippmann, J. Moody and D. Touretzky, Morgan Kaufmann, 1991.

[7] J. H. Schmidhuber, "An $O(N^3)$ Learning Algorithm for Fully Recurrent Networks", Tech Report FKI-151-91, Institut für Informatik, Technische Universität München, May 1991.

[8] Guo-Zheng Sun, Hsing-Hen Chen and Yee-Chun Lee, "A Fast On-line Learning Algorithm for Recurrent Neural Networks", *Proceedings of International Joint Conference on Neural Networks, Seattle, Washington, page II-13*, June 1991.